# Generalisation in Feedforward Networks

**Adam Kowalczyk and Herman Ferra**
Telecom Australia, Research Laboratories
770 Blackburn Road, Clayton, Vic. 3168, Australia
(a.kowalczyk@trl.oz.au, h.ferra@trl.oz.au)

## Abstract

We discuss a model of consistent learning with an additional restriction on the probability distribution of training samples, the target concept and hypothesis class. We show that the model provides a significant improvement on the upper bounds of sample complexity, i.e. the minimal number of random training samples allowing a selection of the hypothesis with a predefined accuracy and confidence. Further, we show that the model has the potential for providing a finite sample complexity even in the case of infinite VC-dimension as well as for a sample complexity below VC-dimension. This is achieved by linking sample complexity to an "average" number of implementable dichotomies of a training sample rather than the maximal size of a shattered sample, i.e. VC-dimension.

## 1 Introduction

A number of fundamental results in computational learning theory [1, 2, 11] links the generalisation error achievable by a set of hypotheses with its Vapnik-Chervonenkis dimension (VC-dimension, for short) which is a sort of capacity measure. They provide in particular some theoretical bounds on the sample complexity, i.e. a minimal number of training samples assuring the desired accuracy with the desired confidence. However there are a few obvious deficiencies in these results: $(i)$ the sample complexity bounds are unrealistically high (c.f. Section 4.), and $(ii)$ for some networks they do not hold at all since VC-dimension is infinite, e.g. some radial basis networks [7].

One may expect that there are at least three main reasons for this state of affairs: (a) that the VC-dimension is too crude a measure of capacity, (b) since the bounds are universal they may be forced too high by some malicious distributions, (c) that particular estimates themselves are too crude, and so might be improved with time. In this paper we will attack the problem along the lines of (a) and (b) since this is most promising. Indeed, even a rough analysis of some proofs of lower bound (e.g. [1]) shows that some of these estimates were determined by clever constructions of discrete, malicious distributions on sets of "shattered samples" (of the size of VC-dimension). Thus this does not necessarily imply that such bounds on the sample complexity are really tight in more realistic cases, e.g. continuous distributions and "non-malicious" target concepts, the point eagerly made by critics of the formalism. The problem is to find such restrictions on target concepts and probability distributions which will produce a significant improvement. The current paper discusses such a proposition which significantly improves the upper bounds on sample complexity.

## 2  A Restricted Model of Consistent Learning

First we introduce a few necessary concepts and some basic notation. We assume we are given *a space of samples* $X$ with *a probability measure* $\mu$, a set $H$ of binary functions $X \mapsto \{0,1\}$ called the *hypothesis space* and *a target concept* $t \in H$. For an $n$-sample $\vec{x} = (x_1, ..., x_n) \in X^n$ and $h : H \to \{0,1\}$ the vector $(h(x_1), ..., h(x_n)) \in \{0,1\}^n$ will be denoted by $h(\vec{x})$. We define two projections $\pi_{\vec{x}}$ and $\pi_{t,\vec{x}}$ of $H$ onto $\{0,1\}^n$ as follows $\pi_{\vec{x}}(h) \overset{def}{=} h(\vec{x}) \overset{def}{=} (h(x_1), ..., h(x_n))$ and $\pi_{t,\vec{x}}(h) \overset{def}{=} \pi_{\vec{x}}(|t-h|) = |t-h|(\vec{x})$ for every $h \in H$. Below we shall use the notation $|S|$ for the cardinality of the set $S$. The average density of the sets of projections $\pi_{\vec{x}}(H)$ or $\pi_{t,\vec{x}}(H)$ in $\{0,1\}^n$ is defined as

$$\mathrm{Pr}_H(\vec{x}) \overset{def}{=} |\pi_{\vec{x}}(H)|/2^n = |\pi_{t,\vec{x}}(H)|/2^n$$

(equivalently, this is the probability of a random vector in $\{0,1\}^n$ belonging to the set $\pi_{\vec{x}}(H)$). Now we define two associated quantities:

$$\bar{\mathrm{Pr}}_{H,\mu}(n) \overset{def}{=} \int \mathrm{Pr}_H(\vec{x})\mu^n(d\vec{x}) = 2^{-n}\int |\pi_{\vec{x}}(H)|\mu^n(d\vec{x}), \qquad (1)$$

$$\mathrm{Pr}_{H,\max}(n) \overset{def}{=} \max_{\vec{x} \in X^n} \mathrm{Pr}_H(\vec{x}).$$

We recall that $d_H \overset{def}{=} \max\{n \ ; \ \exists_{\vec{x} \in X^n} |\pi_{\vec{x}}(H)| = 2^n\}$ is called the Vapnik-Chervonenkis dimension (VC-dimension) of $H$ [1, 11]. If $d_H \leq \infty$ then Sauer's lemma implies the estimates (c.f. [1, 2, 10])

$$\bar{\mathrm{Pr}}_{H,\mu}(n) \leq \mathrm{Pr}_{H,\max}(n) \leq 2^{-n}\Phi(d_H, n) \leq 2^{-n}(en/d_H)^{d_H}, \qquad (2)$$

where $\Phi(d,n) \overset{def}{=} \sum_{i=0}^{d}\binom{n}{i}$ (we assume $\binom{n}{i} \overset{def}{=} 0$ if $i > n$).

Now we are ready to formulate our main assumption in the model. We say that the space of hypotheses $H$ is $(\mu^n, C)$-*uniform* around $t \in 2^X$ if for every set $S \subset \{0,1\}^n$

$$\int |\pi_{t,\vec{x}}(H) \cap S|\mu^n(d\vec{x}) \leq C|S|\bar{\mathrm{Pr}}_{H,\mu}(n). \qquad (3)$$

The meaning of this condition is obvious: we postulate that on average the number of different projections $\pi_{t,\vec{x}}(h)$ of hypothesis $h \in H$ falling into $S$ has a bound proportional to the probability $\bar{\Pr}_{H,\mu}(n)$ of random vector in $\{0,1\}^n$ belonging to the set $\pi_{t,\vec{x}}(H)$. Another heuristic interpretation of (3) is as follows. Imagine that elements of $\pi_{t,\vec{x}}(H)$ are almost uniformly distributed in $\{0,1\}^n$, i.e. with average density $\rho_{\vec{x}} \leq C|\pi_{t,\vec{x}}(H)|/2^n$. Thus the "mass" of the volume $|S|$ is $|\pi_{t,\vec{x}}(H) \cap S| \approx \rho_{\vec{x}}|S|$ and so its average $\int |\pi_{t,\vec{x}}(H) \cap S|\mu^n(d\vec{x})$ has the estimate $\leq |S| \int \rho_{\vec{x}}\mu^n(d\vec{x}) \leq C|S|\bar{\Pr}_{H,\mu}(n)$.

Of special interest is the particular case of *consistent* learning [1], i.e. when the target concept and the hypothesis fully agree on the training sample. In this case, for any $\epsilon > 0$ we introduce the notation

$$Q^\epsilon(m) \stackrel{def}{=} \{\vec{x} \in X^m \; ; \; \exists_{h \in H} \; er_{t,\vec{x}}(h) = 0 \; \& \; er_{t,\mu}(h) \geq \epsilon\},$$

where $er_{t,\vec{x}}(h) \stackrel{def}{=} \sum_{i=1}^m |t-h|(x_i)/m$ and $er_{t,\mu}(h) \stackrel{def}{=} \int |t-h|(x)\mu(dx)$ denote error rates on the training sample $\vec{x} = (x_1, ..., x_m)$ and $X$, respectively. Thus $Q^\epsilon(m)$ is the set of all $m$-samples for which there exists a hypothesis in $H$ with no error on the sample and the error at least $\epsilon$ on $X$.

**Theorem 1** *If the hypothesis space $H$ is $(\mu^{2m}, C)$-uniform around $t \in H$ then for any $\epsilon > 8/m$*

$$\mu^m(Q^\epsilon(m)) \quad \leq \quad \mathcal{E}(m,\epsilon) \stackrel{def}{=} C\bar{\Pr}_{H,\mu}(2m) \sum_{j=\lceil m\epsilon/2 \rceil}^m \binom{2m}{j} 2^{-j} \tag{4}$$

$$\leq \quad C\bar{\Pr}_{H,\mu}(2m)(3/2)^{2m} \leq C(2em/d_H)^{d_H}(3/4)^{2m}. \; \Box \tag{5}$$

Proof of the theorem is given in the Appendix.

Given $\epsilon, \delta > 0$. The integer $m_L(\delta,\epsilon) \stackrel{def}{=} \min\{m > 0 \; ; \; \mu^m(Q^\epsilon) \leq \delta\}$ will be called *the sample complexity* following the terminology of computational learning theory (c.f. [1]). Note that in our case the sample complexity depends also (implicitly) on the target concept $t$, the hypothesis space $H$ and the probability measure $\mu$.

**Corollary 2** *If the hypothesis space $H$ is $(\mu^n, C)$-uniform around $t \in H$ for any $n > 0$, then*

$$m_L(\delta,\epsilon) \quad \leq \quad max\{8/\epsilon, \min\{m \; ; \; 2C\bar{\Pr}_{H,\mu}(2m)(3/2)^m < \delta\}\} \tag{6}$$

$$\leq \quad max\{8/\epsilon, 6.9\,d_H + 2.4\log_2 \frac{C}{\delta}\}. \; \Box \tag{7}$$

The estimate (7) of Corollary 2 reduces to the estimate
$$m_L(\delta,\epsilon) \leq 6.9\,d_H$$
independent of $\delta$ and $\epsilon$. This predicts that under the assumption of the corollary a transition to perfect generalisation occurs for training samples of size $\leq 6.9\,d_H$, which is in agreement with some statistical physics predictions showing such transition occurring below $\approx 1.5 d_H$ for some simple neural networks (c.f. [5]).

**Proof outline.** Estimate (6) follows from the first estimate (5). Estimate (7) can be derived from the second bound in (5) (virtually by repeating the proof of [1, Theorem 8.4.1] with substitution of $4\log_2(4/3)$ for $\epsilon$ and $\delta/C$ for $\delta$). Q.E.D.

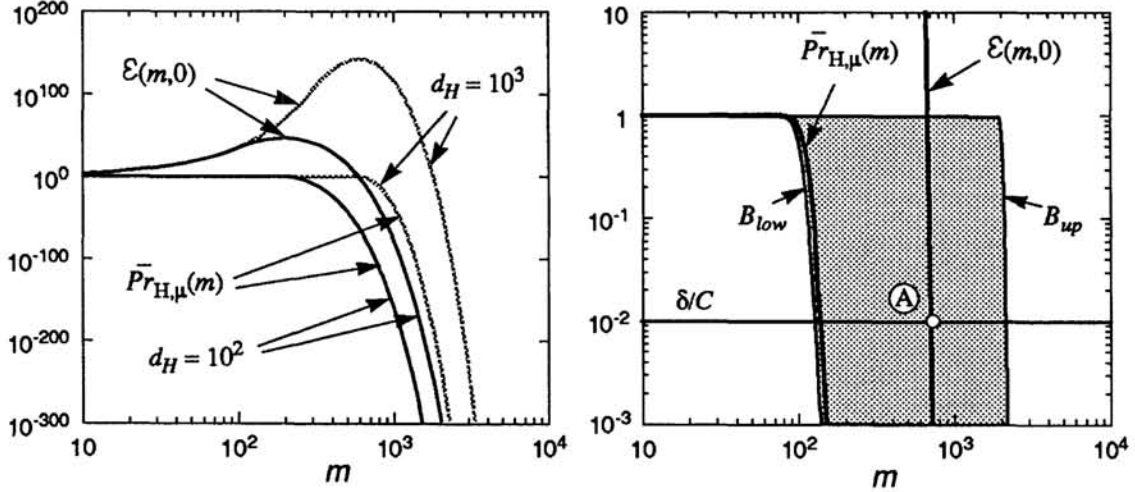

Figure 1: *Plots of estimates on $\bar{P}r_{H,\mu}(m)$ and $\mathcal{E}(m,\epsilon)/C \leq \mathcal{E}(m,0)/C = (3/2)^{2m}\bar{P}r_{H,\mu}(2m)$ (Fig. a) for the analytic threshold neuron ($\bar{P}r_{H,\mu}(m) = \Phi(d_H, 2m)(3/4)^{2m}$ and (Fig. b) for the abstract perceptron according to the estimate (11) on $\bar{P}r_{H,\mu}(m)$ for $m_1 = 50$, $m_2 = 1000$ and $\rho = 0.015$. The upper bound on the sample complexity, $m_L(\epsilon, \delta)$, corresponds to the abscissa value of the intersection of the curve $\mathcal{E}(m,0)$ with the level $\delta/C$ (c.f. point A in Fig.b). In this manner for $\delta/C = 0.01$ we obtain estimates $m_L \leq 602$ and $m_L < 1795$ in the case of Fig. a for $d_H = 100$ and $d_H = 300$ and $m_L \leq 697$ in the case of Fig. b ($d_H = m_2 = 1000$), respectively.*

## 3   An application to feedforward networks

In this section we shall discuss the problem of estimation of $\bar{P}r_{H,\mu}(m)$ which is crucial for application of the above formalism.

First we discuss an example of an *analytic threshold neuron on $R^n$* [6] when $H$ is the family of all functions $\mathbf{R}^n \rightarrow \{0,1\}$, $x \mapsto \theta(\sum_{i=1}^k w_i\alpha_i(x))$, where $\alpha_i : \mathbf{R}^n \rightarrow \mathbf{R}$ are fixed real analytic functions and $\theta$ the ordinary hard threshold. In this case $d_H$ equals the number of linearly independent functions among $\alpha_1, \alpha_2, ..., \alpha_k$. For any continuous probability distribution $\mu$ on $\mathbf{R}^n$ we have:

$$\Pr_H(\vec{x}) = \Phi(d_H, m)/2^m \quad (\forall m \text{ and } \forall \vec{x} \in (\mathbf{R}^n)^m \text{ with probability } 1), \qquad (8)$$

and consequently

$$\bar{P}r_{H,\mu}(m) = \Pr_{H,\max}(m) = \Phi(d_H, m)/2^m \quad \text{for every } m. \qquad (9)$$

Note that this class of neural networks includes as particular cases, *the linear threshold neuron* (if $k = n+1$ and $\alpha_1, ..., \alpha_{n+1}$ are chosen as $1, x_1, ..., x_n$) and *higher order networks* (if $\alpha_i(x)$ are polynomials); in the former case (8) follows also from the classical result of T. Cover [3].

Now we discuss the more complex case of a *linear threshold multilayer perceptron* on $X = \mathbf{R}^n$, with $H$ defined as the family of all functions $\mathbf{R}^n \rightarrow \{0,1\}$ that such an architecture may implement and $\mu$ is any continuous probability measure

on $\mathbf{R}^n$. In this case there exist two functions, $B_{low}(m)$ and $B_{up}(m)$, such that $B_{low}(m) \leq \mathrm{Pr}_H(\vec{x}) \leq B_{up}(m)$ for any $\vec{x} \in (\mathbf{R}^n)^m$ with probability 1. In other words, $\mathrm{Pr}_H(\vec{x})$ takes values within a "bifurcation region" similar to the shaded region in Fig. 1.b. Further, it is known that $B_{low}(m) = 1$ for $m \leq nh_1 + 1$, $B_{up}(m) = 1$ for $m \leq d_H$ and, in general, $B_{low}(m) \leq \Phi(nh_1 + 1, m)$ and $B_{up}(m) \leq \Phi(d_H, m)$, where $h_1$ is the number of neurons in the first hidden layer. Given this we can say that $\bar{\mathrm{Pr}}_{H,\mu}(m)$ takes values somewhere within the "bifurcation region". It is worth noting that the width of the "bifurcation region", which approximately equals $2(d_H - nh_1)$ (since it is known that values on the boundaries have a positive probability of being randomly attained) increases with increasing $h_1$ since [9]

$$\Omega(nh_1 \log_2(h_1)) = \max_p p(n - p)(h_1/2 - 2^p) \leq d_H. \tag{10}$$

Estimate (6) is better than (7) in general, and in particular, if $\bar{\mathrm{Pr}}_{H,\mu}(m)$ "drops" to 0 much quicker than $B_{up} = \Phi(d_H, m)$, we may expect that it will provide an estimate of sample complexity $m_L$ even below $d_H$; if $\bar{\mathrm{Pr}}_{H,\mu}(m)$ is close to $B_{up}$, then the difference between both estimates will be negligible. In order to clarify this issue we shall consider now a third, abstract example.

We introduce the *abstract perceptron* defined as the set of hypotheses $H$ on a probabilistic space $(X, \mu)$ with the following property for a random $m + 1$-tuple $(\vec{x}, x) \in X^m \times X$:

> $d_{VC}(\vec{x}, x) = d_{VC}(\vec{x}) + 1$ with probability $= 1$ if $d_{VC}(\vec{x}) < m_1$ and with probability $\rho$ if $m_1 \leq d_{VC}(\vec{x}) < m_2$, and $d_{VC}(\vec{x}, x) = d_{VC}(\vec{x})$, otherwise.

Here $0 \leq m_1 \leq m_2 \leq \infty$ are two (integer) constants, $0 \leq \rho \leq 1$ is another constant and $d_{VC}(x_1, ..., x_m)$ is the maximal $n$ such that $|\pi_{(x_{i_1}, ..., x_{i_n})}(H)| = 2^n$ for some $1 \leq i_1 < \cdots < i_n \leq m$. It can easily be seen that $d_H = m_2$ in this case and that the threshold analytic neuron is a particular example of abstract perceptron (with $\rho \stackrel{def}{=} 0$ and $m_1 = m_2 = d_H$, $B_{low}(m) = B_{up} = \Phi(m_1, m)/2^m$). Note further, that if $0 < \rho < 1$, then for any $m \geq m_1$, $d_{VC}(\vec{x}) = m_1$ with probability $> 0$, and for any $m \geq m_2$, $d_{VC}(\vec{x}) = d_H = m_2$ with probability $> 0$. In this regard the abstract perceptron resembles the linear threshold multilayer perceptron (with $m_1$ and $m_2$ corresponding to $nh_1 + 1$ and $d_H$, respectively). However, the main advantage of this model is that we can derive the following estimate:

$$\bar{\mathrm{Pr}}_{H,\mu}(m) \leq 2^{-m} \sum_{i=0}^{m-m_1} \binom{m - m_1}{i} \rho^{m-m_1-i}(1 - \rho)^i \Phi(\min(m - i, m_2), m) \tag{11}$$

Using this estimate we find that for sufficiently low $\rho$ (and sufficiently large $m_2$) the sample complexity upper bound (6) is determined by $m_1$ and can even be lower than $m_2 = d_H$ (c.f. Figure 1.b). In particular, the sample complexity determined by Eqns. (6) and (11) can be finite even if $d_H = m_2 = \infty$ (c.f. the curve $\mathcal{E}(m, 0)$ for $\rho = .05$ in Fig. 1.b which is the same for $m_2 = 1000$ and $m_2 = \infty$).

## 4  Discussion

The paper strongly depends on the postulate (3) of $(\mu^n, C)$-uniformity. We admit that this is an ad hoc assumption here as we do not give examples when it is

satisfied nor a method to determine the constant $C$. From this point of view our results at the current stage have no predictive power, perchaps only explanatory one. The paper should be viewed as an attempt in the direction to explain within VC-formalism some known generalisation properties of neural networks which are out of the reach of the formalism to date, such as the empirically observed peak generalisation for backpropagation network trained with samples of the size well below VC-dimension [8] or the phase transitions to perfect generalisation below $1.5.\times$VC-dimension [5]. We see the formalism in this paper as one of a number of possible approaches in this direction. There are other possibilities here as well (e.g. [5, 12]) and in particular other, weaker versions of $(\mu^n, C)$-uniformity can be used leading to similar results. For instance in Theorem 1 and Corollary 2 it was enough to assume $(\mu^n, C)$-uniformity for a special class of sets $S$ ($S = S_{0,j}^{m,\overline{m}}$, c.f. the Appendix); we intend to discuss other options in this regard on another occasion.

Now we relate this research to some previous results (e.g. [2, 4]) which imply the following estimates on sample complexity (c.f. [1, Theorems 8.6.1-2]):

$$\max\left(\frac{d_H - 1}{32\epsilon}, -\ln(\delta)/\epsilon\right) \le m_L^*(\delta, \epsilon) \le \left\lceil \frac{4}{\epsilon}\left(d_H \log_2 \frac{12}{\epsilon} + \log_2 \frac{2}{\delta}\right)\right\rceil, \quad (12)$$

where the lower bound is proved for all $\epsilon \le 1/8$ and $\delta \le 1/100$; here $m_L^*(\delta, \epsilon)$ is the "universal" sample complexity, i.e. for all target concepts $t$ and all probability distributions $\mu$. For $\epsilon = \delta = 0.01$ and $d_H >> 1$ this estimate yields $3d_H < m_L^*(.01, .01) < 4000d_H$. These bounds should be compared against estimates of Corollary 2 of which (7) provides a much tighter upper bound, $m_L(.01, .01) \le 6.9d_H$, if the assumption on $(\mu^m, C)$-uniformity of the hypothesis space around the target concept $t$ is satisfied.

## 5   Conclusions

We have shown that under appropriate restriction on the probability distribution and target concept, the upper bound on sample complexity (and "perfect generalisation") can be lowered to $\approx 6.9\times$ VC-dimension, and in some cases even below VC-dimension (with a strong possibility that multilayer perceptron could be such).

We showed that there are other parameters than VC-dimension potentially impacting on generalisation capabilities of neural networks. In particular we showed by example (abstract perceptron) that a system may have finite sample complexity and infinite VC dimension at the same time.

The formalism of this paper predicts transition to perfect generalisation at relatively low training sample sizes but it is too crude to predict scaling laws for learning curves (c.f. [5, 12] and references in there).

**Acknowledgement.**   The permission of Managing Director, Research and Information Technology, Telecom Australia, to publish this paper is gratefully acknowledged.

## References

[1] M. Anthony and N. Biggs. *Computational Learning Theory*. Cambridge Uni-

versity Press, 1992.

[2] A. Blumer, A. Ehrenfeucht, D. Haussler, and M.K. Warmuth. Learnability and the Vapnik-Chervonenkis dimensions. *Journal of the ACM*, **36**:929–965, (Oct. 1989).

[3] T.M. Cover. Geometrical and statistical properties of linear inequalities with applications to pattern recognition. *IEEE Trans. Elec. Comp.*, EC-14:326–334, 1965.

[4] A. Ehrenfeucht, D. Haussler, M. Kearns, and L. Valiant. A general lower bound on the number of examples needed for learning. *Information and Computation*, **82**:247–261, 1989.

[5] D. Hausler, M. Kearns, H.S. Seung, and N. Tishby. Rigorous learning curve bounds from statistical mechanics. Technical report, 1994.

[6] A. Kowalczyk. Separating capacity of analytic neuron. In *Proc. ICNN'94, Orlando*, 1994.

[7] A. Macintyre and E. Sontag. Finiteness results for sigmoidal "neural" networks. In *Proc. of the 25th Annual ACM Symp. Theory of Comp.*, pages 325–334, 1993.

[8] G.L. Martin and J.A. Pitman. Recognizing handprinted letters and digits using backpropagation learning. *Neural Comput.*, 3:258–267, 1991.

[9] A. Sakurai. Tighter bounds of the VC-dimension of three-layer networks. In *Proceedings of the 1993 World Congress on Neural Networks*, 1993.

[10] N. Sauer. On the density of family of sets. *Journal of Combinatorial Theory (Series A)*, 13:145–147, 1972).

[11] V. Vapnik. *Estimation of Dependences Based on Empirical Data*. Springer-Verlag, 1982.

[12] V. Vapnik, E. Levin, and Y. Le Cun. Measuring the vc-dimension of a learning machine. *Neural Computation*, 6 (5):851–876, 1994).

## 6 Appendix: Sketch of the proof of Theorem 1

The proof is a modification of the proof of [1, Theorem 8.3.1]. We divide it into three stages.

**Stage 1.** Let

$$\mathcal{R}^j \stackrel{def}{=} \{(x,y) \in X^m \times X^m \approx X^{2m} \; ; \; \exists_{h \in H} er_x h = 0 \; \& \; er_y h = j/m\} \qquad (13)$$

for $j \in \{0, 1, ..., m\}$. Using a Chernoff bound on the "tail" of binomial distribution it can be shown [1, Lemma 8.3.2] that for $m \geq 8/\epsilon$

$$\mu^m(Q^\epsilon(m)) \leq 2 \sum_{j \geq \lceil m\epsilon/2 \rceil}^{m} \mu^{2m}(\mathcal{R}^j) \qquad (14)$$

**Stage 2.** Now we use a combinatorial argument to estimate $\mu^{2m}(\mathcal{R}^j)$. We consider the $2^m$-element commutative group $G_m$ of transformations of $X^m \times X^m \approx X^{2m}$ generated by all "co-ordinate swaps" of the form

$$(x_1, ..., x_m, y_1, ..., y_m) \mapsto (x_1, ..., x_{i-1}, y_i, x_{i+1}, ..., x_m, y_1, ..., y_{i-1}, x_i, y_{i+1}, ..., y_m),$$

for $1 \leq i \leq m$. We assume also that $G_m$ transforms $\{0,1\}^m \times \{0,1\}^m \approx \{0,1\}^{2m}$ in a similar fashion. Note that

$$\sigma(\pi_{t,(\vec{x},\vec{y})}(h)) = (\pi_{t,\sigma(\vec{x},\vec{y})}(h) \quad (\text{for } \sigma \in G_m). \tag{15}$$

As transformation $\sigma \in G_m$ preserve the measure $\mu^{2m}$ on $X^m \times X^m$ we obtain

$$2^m \mu^{2m}(\mathcal{R}^j) = |G_m| \mu^{2m}(\mathcal{R}^j) \;\; = \;\; \sum_{\sigma \in G_m} \int \mu^{2m}(d\vec{x}d\vec{y}) \chi_{\mathcal{R}^j}(\sigma(\vec{x},\vec{y}))$$

$$= \int \mu^{2m}(d\vec{x}d\vec{y}) \sum_{\sigma \in G_m} \chi_{\mathcal{R}^j}(\sigma(\vec{x},\vec{y})). \tag{16}$$

Let $S_{0,j}^{m,m} \overset{def}{=} \{\tilde{h} = (\tilde{h}_1, \tilde{h}_2) \in \{0,1\}^m \times \{0,1\}^m \; ; \; \tilde{h}_1 = 0 \; \& \; ||\tilde{h}_2|| = j\}$ and $S_j^{2m} \overset{def}{=} \{\tilde{h} \in \{0,1\}^{2m} \; ; \; ||\tilde{h}|| = j\}$, where $||\tilde{h}|| \overset{def}{=} \tilde{h}_1 + \cdots + \tilde{h}_m$ for any $\tilde{h} = (\tilde{h}_1, ..., \tilde{h}_m) \in \{0,1\}^m$. Then $|S_j^{2m}| = \binom{2m}{j}$, $\sigma(S_{0,j}^{m,m}) \subset S_j^{2m}$ for any $\sigma \in G_m$ and

$$\mathcal{R}^j = \{(\vec{x},\vec{y}) \in X^m \times X^m \; ; \; \exists \tilde{h} \in \pi_{t,(\vec{x},\vec{y})}(H) \cap S_{0,j}^{m,m}\}, \tag{17}$$

Thus from Eqn. (16) we obtain

$$2^m \mu^{2m}(\mathcal{R}^j) \;\; \leq \;\; \int \mu^{2m}(d\vec{x}d\vec{y}) \sum_{\sigma \in G_m} \sum_{\tilde{h} \in \pi_{t,(\vec{x},\vec{y})}(H) \cap S_j^{2m}} \chi_{S_{0,j}^{m,m}}(\sigma\tilde{h})$$

$$= \int \mu^{2m}(d\vec{x}d\vec{y}) \sum_{\tilde{h} \in \pi_{t,(\vec{x},\vec{y})}(H) \cap S_j^{2m}} \sum_{\sigma \in G_m} \chi_{S_{0,j}^{m,m}}(\sigma\tilde{h})$$

$$= \int \mu^{2m}(d\vec{x}d\vec{y}) \sum_{\tilde{h} \in \pi_{t,(\vec{x},\vec{y})}(H) \cap S_{0,j}^{m,m}} |\{\sigma \in G_m \; ; \; \sigma\tilde{h} \in S_j^{2m}\}|$$

$$= \int \mu^{2m}(d\vec{x}d\vec{y}) |\{\pi_{t,(\vec{x},\vec{y})}(H) \cap S_j^{2m}\}| \, 2^{m-j}.$$

Applying now the condition of $(\mu^{2m}, C)$-uniformity (Eqn. 3), Eqn. 17 and dividing by $2^m$ we get

$$\mu^{2m}(\mathcal{R}^j) \leq C \bar{\mathrm{Pr}}_{H,\mu}(2m) \binom{2m}{j} 2^{-j}.$$

**Stage 3.** On substitution of the above estimate into (14) we obtain estimate (4). To derive (5) let us observe that $\sum_{j=\lceil m\epsilon/2 \rceil}^{m} \binom{2m}{j} 2^{-j} \leq (1 + 1/2)^{2m}$. Q.E.D.